# Connectionist Speaker Normalization with Generalized Resource Allocating Networks

**Cesare Furlanello**
Istituto per La Ricerca
Scientifica e Tecnologica
Povo (Trento), Italy
`furlan@irst.it`

**Diego Giuliani**
Istituto per La Ricerca
Scientifica e Tecnologica
Povo (Trento), Italy
`giuliani@irst.it`

**Edmondo Trentin**
Istituto per La Ricerca
Scientifica e Tecnologica
Povo (Trento), Italy
`trentin@irst.it`

## Abstract

The paper presents a rapid speaker-normalization technique based on neural network spectral mapping. The neural network is used as a front-end of a continuous speech recognition system (speaker-dependent, HMM-based) to normalize the input acoustic data from a new speaker. The spectral difference between speakers can be reduced using a limited amount of new acoustic data (40 phonetically rich sentences). Recognition error of phone units from the acoustic-phonetic continuous speech corpus APASCI is decreased with an adaptability ratio of 25%. We used local basis networks of elliptical Gaussian kernels, with recursive allocation of units and on-line optimization of parameters (*GRAN* model). For this application, the model included a linear term. The results compare favorably with multivariate linear mapping based on constrained orthonormal transformations.

## 1   INTRODUCTION

Speaker normalization methods are designed to minimize inter-speaker variations, one of the principal error sources in automatic speech recognition. Training a speech recognition system on a particular speaker (speaker-dependent or SD mode) generally gives better performance than using a speaker-independent system, which is

trained to recognize speech from a generic user by averaging over individual differences. On the other hand, performance may be dramatically worse when a SD system "tailored" on the acoustic characteristics of a speaker (the *reference* speaker) is used by another one (the *new* or *target* speaker). Training a SD system for any new speaker may be unfeasible: collecting a large amount of new training data is time consuming for the speaker and unacceptable in some applications. Given a pre-trained SD speech recognition system, the goal of normalization methods is then to reduce to a few sentences the amount of training data required from a new speaker to achieve acceptable recognition performance. The inter-speaker variation of the acoustic data is reduced by estimating a feature vector transformation between the acoustic parameter space of the new speaker and that of the reference speaker (Montacie et al., 1989; Class et al., 1990; Nakamura and Shikano, 1990; Huang, 1992; Matsukoto and Inoue, 1992). This multivariate transformation, also called *spectral mapping* given the type of features considered in the parameterization of speech data, provides an acoustic front-end to the recognition system. Supervised speaker normalization methods require that the text of the training utterances required from the new speaker is known, while arbitrary utterances can be used by unsupervised methods (Furui and Sondhi, 1991). Good performance have been achieved with spectral mapping techniques based on MSE optimization (Class et al., 1990; Matsukoto and Inoue, 1992). Alternative approaches presented estimation of the spectral normalization mapping with Multi-Layer Perceptron neural networks (Montacie et al., 1989; Nakamura and Shikano, 1990; Huang, 1992; Watrous, 1994).

This paper introduces a supervised speaker normalization method based on neural network regression with a generalized local basis model of elliptical kernels (Generalized Resource Allocating Network: *GRAN* model). Kernels are recursively allocated by introducing the heuristic procedure of (Platt, 1991) within the generalized RBF schema proposed in (Poggio and Girosi, 1989). The model includes a linear term and efficient on-line optimization of parameters is achieved by an automatic differentiation technique. Our results compare favorably with normalization by affine linear transformations based on orthonormal constrained pseudoinverse. In this paper, the normalization module was integrated and tested as an acoustic front-end for speaker-dependent continuous speech recognition systems. Experiments regarded phone units recognition with Hidden Markov Model (HMM) recognition systems.

The diagram in Figure 1 outlines the general structure of the experiment with *GRAN* normalization modules. The architecture is independent from the specific speech recognition system and allows comparisons between different normalization techniques. The *GRAN* model and a general procedure for data standardization are described in Section 2 and 3. After a discussion of the spectral mapping problem in Section 4, the APASCI corpus used in the experiments and the characteristics of the acoustic data are described in Section 5. The recognition system and the experiment set-up are detailed in Sections 6–8. Results are presented and discussed in Section 9.

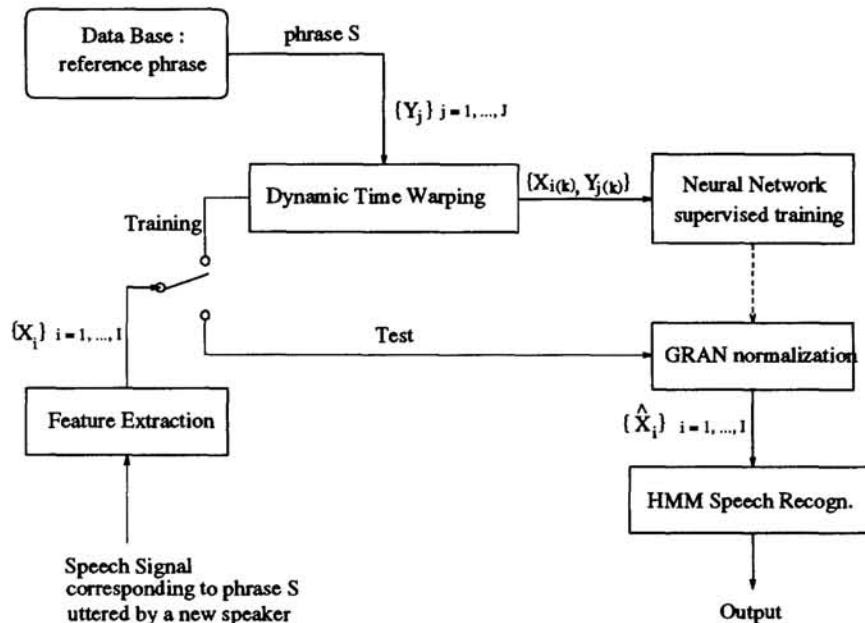

Figure 1: System overview

## 2   THE GRAN MODEL

Feedforward artificial neural networks can be regarded as a convenient realization of general functional superpositions in terms of simpler kernel functions (Barron and Barron, 1988). With one hidden layer we can implement a multivariate superposition $f(\boldsymbol{x}) = \sum_{j=0}^{N} \alpha_j K_j(\boldsymbol{x}, \boldsymbol{\omega}_j)$ where $K_j$ is a function depending on an input vector $\boldsymbol{x}$ and a parameter vector $\boldsymbol{\omega}_j$, a general structure which allows to realize flexible models for multivariate regression. We are interested in the schema: $\hat{\mathbf{y}} = HK(\mathbf{x}) + A\mathbf{x} + \mathbf{b}$ with input vector $\mathbf{x} \in \mathbf{R}^{d_1}$ and estimated output vector $\hat{\mathbf{y}} \in \mathbf{R}^{d_2}$. $K = (K_j)$ is a $n$-dimensional vector of local kernels, $H$ is the $d_2 \times n$ real matrix of kernel coefficients, $\mathbf{b} \in \mathbf{R}^{d_2}$ is an offset term and $A$ is a $d_2 \times d_1$ linear term. Implemented kernels are Gaussian, Hardy multiquadrics, inverse of Hardy multiquadrics and Epanenchnikov kernels, also in the Nadaraya-Watson normalized form (Härdle, 1990). The kernel allocation is based on a recursive procedure: if appropriate novelty conditions are satisfied for the example $(\mathbf{x}', \mathbf{y}')$, a new kernel $K_{n+1}$ is allocated and the new estimate $\hat{\mathbf{y}}_{n+1}$ becomes $\hat{\mathbf{y}}_{n+1}(\mathbf{x}) = \hat{\mathbf{y}}_n(\mathbf{x}) + K_{n+1}(\|\mathbf{x} - \mathbf{x}'\|_w)(\mathbf{y}' - \hat{\mathbf{y}}_n(\mathbf{x}))$ (Härdle, 1990). Global properties and rates of convergence for recursive kernel regression estimates are given in (Krzyzak, 1992). The heuristic mechanism suggested by (Platt, 1991) has been extended to include the optimization of the weighted metrics as requested in the generalized versions of RBF networks of (Poggio and Girosi, 1989). Optimization regards kernel coefficients, locations and bandwidths, the offset term, the coefficient matrix $A$ if considered, and the $W$ matrix defining the weighted metrics in the input space: $\|\mathbf{x}\|_W^2 = \mathbf{x}^t W^t W \mathbf{x}$. Automatic differentiation is used for efficient on-line gradient-descent procedure w.r.t. different error functions ($L_2$, $L_1$, entropy fit), with different learning rates for each type of parameters.

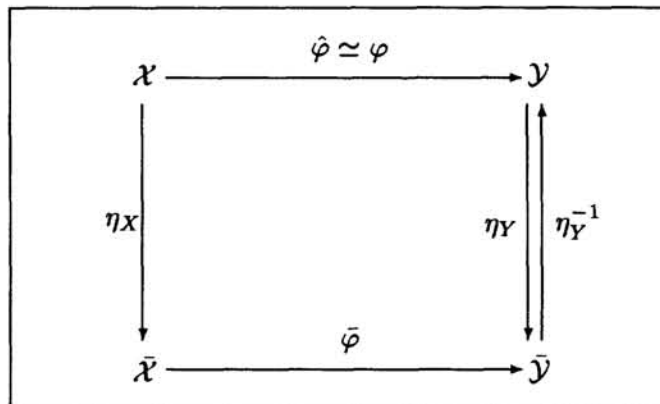

Figure 2: Commutative diagram for the speaker normalization problem. The spectral mapping $\varphi$ between original spaces $\mathcal{X}$ and $\mathcal{Y}$ is estimated by $\hat{\varphi} = \eta_Y^{-1} \cdot \bar{\varphi} \cdot \eta_X$, obtained by composition of the neural GRAN mapping $\bar{\varphi}$ between PCA spaces $\bar{\mathcal{X}}$ and $\bar{\mathcal{Y}}$ with the two invertible PCA transformations $\eta_X$ and $\eta_Y$.

## 3   NETWORKS AND PCA TRANSFORMATIONS

The normalization module is designed to estimate a spectral mapping between the acoustic spaces of two different speakers. Inter-speaker variability is reflected by significant differences in data distribution in these multidimensional spaces (we considered 8 dimensions); in particular it is important to take into account *global* data anisotropy. More generally, it is also crucial to decorrelate the features describing the data. A general recipe is to apply the well-known Principal Component Analysis (PCA) to the data, in this case implemented from standard numerical routines based on Singular Value Decomposition of the data covariance matrices. The network was applied to perform a mapping between the new feature spaces obtained from the PCA transformations, mean translation included (Figure 2).

## 4   THE SPECTRAL MAPPING PROBLEM

A sound uttered by a speaker is generally described by a sequence of feature vectors obtained from the speech signal via short-time spectral analysis (Sec. 5). The spectral representations of the same sequence of sounds uttered by two speakers are subject to significant variations (e.g. differences between male and female speakers, regional accents, . . . ). To deal with acoustic differences, a suitable transformation (the *spectral mapping*) is sought which performs the "best" mapping between the corresponding spectra of two speakers. Let $Y = (y_1, y_2, ..., y_J)$ and $X = (x_1, x_2, ..., x_I)$ be the spectral feature vector sequences of the same sentence uttered by two speakers, called respectively the *reference* and the *new* speaker. The desired mapping is performed by a function $\varphi(x_i)$ such that the transformed vector sequence obtained from $X = (x_i)$ approximates as close as possible the spectral vector sequence $Y = (y_j)$. To eliminate time differences between the two acoustic realizations, a *time warping* function has to be determined yielding pairs $C(k) = (i(k), j(k))_{k=1...K}$ of corresponding indexes of feature vectors in $X$ and $Y$, respectively. The desired spectral

mapping $\varphi(x_i)$ is the one which minimizes $\sum_{k=1}^{K} d(y_{j(k)}, \varphi(x_{i(k)}))$ where $d(\cdot, \cdot)$ is a distorsion measure in the acoustic feature space. To estimate the transformation, a set of supervised pairs $(x_{i(k)}, y_{j(k)})$ is considered. In summary, the training material considered in the experiments consisted of a set of vector pairs obtained by applying the Dynamic Time Warping (DTW) algorithm (Sakoe and Chiba, 1978) to a set of phrases uttered by the reference and the new speaker.

## 5   THE APASCI CORPUS

The experiments reported in this paper were performed on a portion of APASCI, an italian acoustic-phonetic continuous speech corpus. For each utterance, text and phonetic transcriptions were automatically generated (Angelini et al., 1994). The corpus consists of two portions. The first part, for the training and validation of speaker independent recognition systems, consists of a training set (2140 utterances), a development set (900 utterances) and a test set (860 utterances). The sets contain, respectively, speech material from 100 speakers (50 males and 50 females), 36 speakers (18 males and 18 females) and 40 speakers (20 males and 20 females). The second portion of the corpus is for training and validation of speaker dependent recognition systems. It consists of speech material from 6 speakers (3 males and 3 females). Each speaker uttered 520 phrases, 400 for training and 120 for test. Speech material in the test set was acquired in different days with respect to the training set. A subset of 40 utterances from the training material forms the adaptation training set, to be used for speaker adaptation/normalization purposes.

For this application, each signal in the corpus was processed to obtain its parametric representation. The signal was preemphasized using a filter with transfer function $H(z) = 1 - 0.95 \times z^{-1}$, and a 20 ms Hamming window is then applied every 10 ms. For each frame, the normalized log-energy as well as 8 Mel Scaled Cepstral Coefficients (MSCC) based on a 24-channel filter-bank were computed. Normalization of log-energy was performed by subtracting the maximum log-energy value in the sentence; for each Mel coefficient, normalization was performed by subtracting the mean value of the whole utterance. For both MSCC and the log-energy, the first order derivatives as well as the second order derivatives were computed. For each frame, all the computed acoustic parameters were combined in a single feature vector with 27 components.

## 6   THE RECOGNITION SYSTEM

For each of the 6 speakers, a SD HMM recognition system was trained with the 400 utterances available in the APASCI corpus; the systems were bootstrapped with gender dependent models trained on the gender dependent speech material (1000 utterances for male and 1140 utterances for female). A set of 38 context independent acoustic-phonetic units was considered. Left-to-right HMMs with three and four states were adopted for short (i.e. $p, t, k, b, d, g$) and long (e.g. $a, i, u, o, e$) sounds respectively. Silence, pause and breath were modeled with a single state ergodic model. The output distribution probabilities were modeled with mixtures of 16 gaussian probability densities, diagonal covariance matrixes. Transitions leaving the same state shared the same output distribution probabilities.

|        | anco0 | bian0 | ilco0 | dafa0 | gisv0 | saor0 |
|--------|-------|-------|-------|-------|-------|-------|
| anco0  | 82.75 | 76.89 | 71.37 | 69.38 | 64.99 | 59.89 |
| bian0  | 68.80 | 86.91 | 65.70 | 62.77 | 58.27 | 50.86 |
| ilco0  | 62.71 | 66.32 | 85.02 | 64.26 | 64.53 | 62.38 |
| dafa0  | 57.74 | 56.88 | 56.84 | 85.83 | 68.01 | 70.21 |
| gisv0  | 62.43 | 64.32 | 68.19 | 76.20 | 83.44 | 71.37 |
| saor0  | 54.86 | 53.07 | 61.77 | 76.03 | 69.64 | 88.21 |

Table 1: Phone Recognition Rate (Unit Accuracy %) without normalization

# 7    TRAINING THE NORMALIZATION MODULES

A set of 40 phrases was considered for each pair (*new, reference*) of speakers to train the normalization modules. In order to take into account alternative pronunciation, insertion or deletion of phonemes, pauses between words and other phenomena, the automatic phonetic transcription and segmentation available in APASCI was used for each utterance. Given two utterances corresponding to the same phrase, we considered only their segments having the same phonetic transcription. To determine these segments the DTW algorithm was applied to the phonetic transcription of the two utterances. The DTW algorithm was applied a second time to the obtained segments and the resulting *optimal* alignment paths gave the desired set of vector pairs. The DTW algorithm was applied only to the 8 MSCC and the other acoustic parameters were left unmodified.

We trained networks with 8 inputs and 8 outputs. The model included a linear term: first the linear term was fit to the data, and then the rest of the expansion was estimated by fitting the residuals of the linear regression. The networks grew up to 50 elliptical gaussian kernels using dynamic allocation. Kernel coefficients, locations and bandwidths were optimized using different learning rates for 10 epochs w.r.t the $L_1$ norm, which proved to be more efficient than the usual $L_2$ norm.

# 8    THE RECOGNITION EXPERIMENTS

Experiments concerned continuous phone recognition without any lexical and phonetical constraint (no phone statistic was used). For all the couples (*new, reference*) of speakers in the database, a recognition experiment was performed using 90 (of the 120 available) test utterances from the new speaker with the SD recognition system previously trained for the reference speaker. On average the test sets consisted of 4770 phone units. The experiments were repeated transforming the test data with different normalization modules and performance compared. Results are expressed in terms of insertions (*Ins*), deletions (*Del*) and substitutions (*Sub*) of phone units made by the recognizer. Unit Accuracy (*UA*) and Percent Correct (*PC*) performance indicators are respectively defined w.r.t. the total number of units $n_{units}$ as $UA = 100\,(1 - (Ins + Del + Sub)/n_{units})$ and $PC = 100\,(1 - (Del + Sub)/n_{units})$ . In Table 1 the baseline speaker dependent performance for the 6 speaker dependent systems is reported. Row labels indicate the speaker reference model while column labels identify whose target acous-

|        | anco0 | bian0 | ilco0 | dafa0 | gisv0 | saor0 |
|--------|-------|-------|-------|-------|-------|-------|
| anco0  | –     | 79.11 | 73.88 | 72.57 | 69.79 | 69.86 |
| bian0  | 71.28 | –     | 70.89 | 71.23 | 67.04 | 67.72 |
| ilco0  | 65.21 | 70.52 | –     | 67.64 | 66.56 | 66.84 |
| dafa0  | 63.66 | 66.88 | 63.85 | –     | 70.54 | 74.80 |
| gisv0  | 66.07 | 69.90 | 71.29 | 78.01 | –     | 75.32 |
| saor0  | 61.88 | 67.03 | 66.21 | 76.89 | 70.44 | –     |

Table 2: Phone Recognition Rate (Unit Accuracy %) with NN normalization

tic data are used. Thus $UA$ and $PC$ entries in the main diagonal are for the same speaker who trained the system while the remaining entries relate to performance obtained with new speakers. We also considered the *adaptability ratios* for $a = UA$ and $p = PC$ (Montacie et al., 1989): $\rho_a = (a_{RT}^n - a_{RT})/(a_{RR} - a_{RT})$ and $\rho_p = (p_{RT}^n - p_{RT})/(p_{RR} - p_{RT})$ where $a_{RT}$ indicate accuracy for reference speaker $R$ and target $T$ without normalization, $a_{RR}$ is the speaker dependent baseline accuracy and apex $n$ indicates normalization. The same notation applies to the percent correct adaptability ratio $\rho_p$.

# 9   RESULTS AND CONCLUSIONS

Normalization experiments have been performed with the set-up described in the previous Section. The phone recognition rates obtained with normalization modules based on the GRAN model are reported in Table 2 in terms of Unit Accuracy (dee Table 1 for the baseline performance). In Table 3 the performance of the GRAN model (NN) and constrained orthonormal linear mapping (LIN) are compared with the baseline performance (SD: no adaptation) in terms of both Unit Accuracy and Percent Correct. The network shows an improvement, as evidenced by the variation in the $\rho_a$ and $\rho_p$ values. Results are reported averaging performance over all the pairs (new,reference) of speakers (Total column), and considering pairs of speakers of the same gender and of different genders (*Female*: only female subjects, *Male*: only males, *Diff*: different genders). An analysis of the adaptability ratios shows that the effect of the network normalization is higher than with the linear network for all the 3 subgroups of pairs: $\rho_a^{NN} = 0.20$ vs $\rho_a^{LIN} = 0.16$ for the Female couples and $\rho_a^{NN} = 0.16$ vs $\rho_a^{LIN} = 0.15$ for the Male couples. The improvement is higher ($\rho_a^{NN} = 0.28, \rho_a^{LIN} = 0.24$) for speaker of different genders. Although these preliminary experiments show only a minor improvement of performance achieved by the network with respect to linear mappings, we expect that the selectivity of the network could be exploited using acoustic contexts and code dependent neural networks.

**Acknowledgements**

This work has been developed within a grant of the "Programma Nazionale di Ricerca per la Bioelettronica" assigned by the Italian Ministry of University and Technologic Research to Elsag Bailey. The authors would like to thank B. Angelini, F. Brugnara, B. Caprile, R. De Mori, D. Falavigna, G. Lazzari and P. Svaizer.

|      |     | Total | Female | Male  | Diff  | $\rho$ |
|------|-----|-------|--------|-------|-------|--------|
| SD   | UA  | 64.56 | 68.63  | 71.91 | 60.75 | -      |
|      | PC  | 70.87 | 74.35  | 77.55 | 67.48 | -      |
| LIN  | UA  | 69.04 | 71.20  | 73.97 | 66.67 | 0.21   |
|      | PC  | 74.97 | 76.79  | 79.27 | 72.93 | 0.23   |
| NN   | UA  | 69.76 | 71.81  | 74.30 | 67.56 | 0.25   |
|      | PC  | 75.59 | 77.27  | 79.55 | 73.71 | 0.28   |

Table 3: Phone Recognition Rate (%) in terms of both Unit Accuracy, Percent Correct, and adaptability ratio $\rho$.

# References

Angelini, B., Brugnara, F., Falavigna, D., Giuliani, D., Gretter, R., and Omologo, M. (September 1994). Speaker Independent Continuous Speech Recognition Using an Acoustic-Phonetic Italian Corpus. In *Proc. of ICSLP*, pages 1391–1394.

Barron, A. R. and Barron, R. L. (1988). Statistical learning networks: a unifying view. In *Symp. on the Interface: Statistics and Computing Science*, Reston, VI.

Class, F., Kaltenmeier, A., Regel, P., and Troller, K. (1990). Fast speaker adaptation for speech recognition system. In *Proc. of ICASSP 90*, pages I–133–136.

Furui, S. and Sondhi, M. M., editors (1991). *Advances in Speech Signal Processing*. Marcel Dekker and Inc.

Härdle, W. (1990). *Applied nonparametric regression*, volume 19 of *Econometric Society Monographs*. Cambridge University Press, New York.

Huang, X. D. (1992). Speaker normalization for speech recognition. In *Proc. of ICASSP 92*, pages I–465–468.

Krzyzak, A. (1992). Global convergence of the recursive kernel regression estimates with applications in classification and nonlinear system estimation. *IEEE Transactions on Information Theory*, 38(4):1323–1338.

Matsukoto, H. and Inoue, H. (1992). A piecewise linear spectral mapping for supervised speaker adaptation. In *Proc. of ICASSP 92*, pages I–449–452.

Montacie, C., Choukri, K., and Chollet, G. (1989). Speech recognition using temporal decomposition and multi-layer feed-forward automata. In *Proc. of ICASSP 89*, pages I–409–412.

Nakamura, S. and Shikano, K. (1990). A comparative study of spectral mapping for speaker adaptation. In *Proc. of ICASSP 90*, pages I–157–160.

Platt, J. (1991). A resource-allocating network for function interpolation. *Neural Computation*, 3(2):213–225.

Poggio, T. and Girosi, F. (1989). A theory of networks for approximation and learning. A.I. Memo No. 1140, MIT.

Sakoe, H. and Chiba, S. (1978). Dynamic programming algorithm optimization for spoken word recognition. *IEEE-ASSP*, 26(1):43–49.

Watrous, R. (1994). Speaker normalization and adaptation using second-order connectionist networks. *IEEE Trans. on Neural Networks*, 4(1):21–30.
